# GRAPHTRAIL: Translating GNN Predictions into Human-Interpretable Logical Rules

**Burouj Armgaan, Manthan Dalmia**
Department of Computer Science & Engineering
IIT Delhi, India
csz228001@iitd.ac.in,
manthandalmia2@gmail.com

**Sourav Medya**
Department of Computer Science
University of Illinois, Chicago, USA
medya@uic.edu

**Sayan Ranu**
Department of Computer Science & Engineering and Yardi School of AI
IIT Delhi, India
sayanranu@cse.iitd.ac.in

## Abstract

Instance-level explanation of graph neural networks (GNNs) is a well-studied area. These explainers, however, only explain an instance (e.g., a graph) and fail to uncover the combinatorial reasoning learned by a GNN from the training data towards making its predictions. In this work, we introduce GRAPHTRAIL, the first *end-to-end*, post-hoc, global GNN explainer that translates the functioning of a black-box GNN model to a boolean formula over the (sub)graph-level concepts without relying on local explainers. GRAPHTRAIL is unique in automatically mining the discriminative subgraph-level concepts using *Shapley values*. Subsequently, the GNN predictions are mapped to a human-interpretable boolean formula over these concepts through *symbolic regression*. Extensive experiments across diverse datasets and GNN architectures demonstrate significant improvement over existing global explainers in mapping GNN predictions to faithful logical formulae. The robust and accurate performance of GRAPHTRAIL makes it invaluable for improving GNNs and facilitates adoption in domains with strict transparency requirements.

## 1 Introduction and Related Works

GNNs have witnessed widespread adoption for graph-level prediction tasks due to their impressive performance [35, 14, 46, 11, 41, 18, 38, 32, 5, 12, 53, 22]. Unfortunately, like other deep-learning models, GNNs are considered black boxes due to their lack of transparency and interpretability. This lack of interpretability presents a significant barrier to their adoption in critical domains such as healthcare, finance, and law enforcement. Additionally, the ability to explain predictions is crucial for understanding potential flaws in the model and generating insights for further refinement.

**Existing Works:** To introduce interpretability of GNNs, several algorithms have been proposed in the literature [23]. Fig. G in the Appendix, which was originally presented in [23], and now updated by us with more recent works, presents the taxonomy of GNN explainability research. As observed, a vast majority of explainers focus on *instance-level* explanations.

Instance-level (or local) explainers [55, 29, 42, 58, 15, 57, 27, 45, 25, 3, 1, 51, 47, 26, 6, 19] take a graph as input and identify components within this graph—-such as a subgraph—that maximally influence the prediction made by a model. This instance-level focus limits its ability to extract patterns utilized by GNNs at a global level across a multitude of graphs and how these patterns are combined into a single decision-making rule. The objective of our work is to develop an end-to-end

global explainer that *(1)* mines the subgraph concepts used by a black-box GNN model, and then *(2)* uncovers the boolean logic used by the GNN over these concepts to make its predictions.

Works on global GNN explainers are limited [56, 16, 54, 2]. XGNN [56] and GNNInterpreter [50] are generative-modeling based global explainers. Both generate a graph that maximally aligns with a specified class label. While this generated graph likely contains important features used by the GNN in making its predictions, it may not actually be present in the dataset, limiting its utility for analyzing specific predictions. Additionally, it does not produce a human-interpretable rule explaining the class attributions made by the GNN. Finally, XGNN [56] requires domain-specific validity-rules as input, which affects its generalizability and results in inferior performance [2]. Another work [54] evaluates which base *concepts* are detected by the neurons of a GNN model during predictions and their relative importance. These concepts can be subgraphs or node-level properties, such as degrees. However, this method lacks the ability to automatically mine the concepts. More importantly, this method also does not generate a human-interpretable rule for the decision-making process of the GNN.

The closest work to ours is GLGEXPLAINER [2], which shares the objective of providing a global explanation of the GNN through a boolean formula over subgraph-level concepts. However, there are significant limitations that need to be addressed:

- **Dependency on instance-level explainers:** GLGEXPLAINER does not mine the subgraph concepts in an end-to-end manner. Instead, it relies on an instance-level explainer (e.g. PGEXPLAINER [29]) to provide these concepts, over which it searches for the combinatorial formula mapping to the GNN predictions. This dependency creates a disconnect with the objective, as instance-level explainers lack a global understanding of the model. Our proposed approach develops an end-to-end pipeline that mines concepts based on global trends.
- **Lack of interpretability due to vector-level concepts:** In GLGEXPLAINER, each concept in the formula corresponds to a feature vector and not a subgraph. These vectors represent the embedding of a cluster of subgraphs generated by the instance explainer. Hence, in its original form, the formula is not human-interpretable. To convert into a human-interpretable formula, GLGEXPLAINER randomly selects a subgraph from the cluster, assuming all subgraphs in a cluster are similar. Our investigation (§ 4) reveals that this assumption is rarely true in practice, compromising both interpretability and efficacy.
- **Lack of robustness:** GLGEXPLAINER shows significant variation in the formula based on the training split used. As our analysis in § 4 reveals, due to the reliance on instance-level explanations, even when data are drawn from the same distribution, the base concept candidates vary, and consequently so does the eventual formula.

At this juncture, we note that our work is distinct from the line of research on explainable GNNs [61, 10, 34]. Explainable GNNs are designed to make explainable predictions rather than explaining the predictions of a black-box GNN.

**Contributions:** In this work, we present an end-to-end, post-hoc, global GNN explainer called GRAPHTRAIL (TRAnslating GNN Prediction into human-Interpretable Logical Rules)[1], which addresses the limitations outlined above. Specifically,

- **Problem formulation:** We formulate the problem of translating a message-passing GNN model for graph classification into a human-interpretable logic formula over subgraph concepts. Unlike existing works, in our formulation, the concepts are not assumed to be an input generated through a decoupled algorithm.
- **Novel methodology:** We develop GRAPHTRAIL, which uses a mix of several innovative insights. First, GRAPHTRAIL exploits the fact that a message passing GNN decomposes a graph into a set of computation trees. This enables GRAPHTRAIL to limit the exploration of concepts from an exponential subgraph search space to a linear space of computation trees. The global impact of computation trees is assessed using *Shapley values*, and then mapped to a boolean formula over concepts using *symbolic regression*.
- **Empirical analysis:** Extensive experiments across a diverse set of datasets, GNN architectures and pooling function, demonstrate GRAPHTRAIL to significantly surpass existing global explainers in efficacy, human-interpretability, data efficiency, and robustness.

## 2 Preliminaries and Problem Formulation

**Definition 1** (Graph). *A graph is defined as $\mathcal{G} = (\mathcal{V}, \mathcal{E}, \boldsymbol{X})$ over a node set $\mathcal{V}$, edge set $\mathcal{E} = \{(u, v) \mid u, v \in \mathcal{V}\}$ and a node feature matrix $\boldsymbol{X} = \{\mathbf{x}_v \mid v \in \mathcal{V}\}$ where $\mathbf{x}_v \in \mathbb{R}^d$ is the set of features characterizing each node.*

Two graphs are termed identical if they are *isomorphic* to each other.

**Definition 2** (Graph Isomorphism). *Graph $\mathcal{G}_1$ is isomorphic to graph $\mathcal{G}_2$ (denoted as $\mathcal{G}_1 \cong \mathcal{G}_2$) if there exists a bijection between their node sets that preserves the edge connectivity and node features. Specifically, $\mathcal{G}_1 \cong \mathcal{G}_2 \iff \exists f : \mathcal{V}_1 \to \mathcal{V}_2$ such that: (1) $f$ is a bijection, (2) $\mathbf{x}_v = \mathbf{x}_{f(v)}$, where $\mathbf{x}_v \in \mathcal{V}_1, \mathbf{x}_{f(v)} \in \mathcal{V}_2$ and (3) $(u, v) \in \mathcal{E}_1$ if and only if $(f(u), f(v)) \in \mathcal{E}_2$.*

Graph $\mathcal{G}_1$ is *subgraph isomorphic* to $\mathcal{G}_2$, denoted as $\mathcal{G}_1 \subseteq \mathcal{G}_2$, if $f$ is an *injection* and condition (3) is modified to $(u, v) \in \mathcal{E}_1$ if $(f(u), f(v)) \in \mathcal{E}_2$.

**Definition 3** (Graph Classification). *In graph classification, we are given a set of train graphs $\mathcal{D}_{tr} = \{\mathcal{G}_1, \cdots, \mathcal{G}_m\}$, where each graph $\mathcal{G}_i$ is tagged with a class label $\mathcal{Y}_i$ from the set $\{\mathcal{Y}_1, \cdots, \mathcal{Y}_c\}$. The objective is to train a GNN model $\Phi$ such that given a graph with an unknown class label, the label prediction error is minimized.*

Error may be measured using any of the known metrics such as cross-entropy loss, negative log-likelihood, etc. Hereon, we implicitly assume $\Phi$ to be a *message-passing* GNN [22, 14, 46, 53]. We assume the GNN $\Phi$ returns a $c$-dimensional distribution over the class labels, where $\Phi(\mathcal{G})_j$ is the probability of the $j$-th class, and $\Phi(\mathcal{G})^* = \arg\max_{j \in \{1,2,\cdots,c\}}\{\Phi(\mathcal{G})_j\}$ denotes the predicted class label.

**Definition 4** (Concepts [2, 21]). *Concepts refer to semantically meaningful units of information within the data that humans use to analyze and make decisions about that dataset. In the context of graph classification, these concepts correspond to subgraphs.*

**Problem 1** (Global GNN explanation through boolean logic over concepts). *Let $\mathcal{D}$ be a set of graphs, where each graph is labeled with a class from the set $\{\mathcal{Y}_1, \cdots, \mathcal{Y}_c\}$. Given a trained GNN $\Phi$, our objective is to learn a set of $c$ boolean formulas $\{f_1, \cdots, f_c\}$, over subgraph-level concepts such that for any graph $\mathcal{G}$, if $\Phi(\mathcal{G})^* = \mathcal{Y}_i$, then $f_i(\mathcal{G}) = $ `TRUE` and $\forall j \neq i$, $f_j(\mathcal{G}) = $ `FALSE`. The candidate space of concepts includes all unique subgraphs of the dataset, i.e., $\mathcal{C} = \{\mathcal{S} \mid \mathcal{S} \subseteq \mathcal{G}, \mathcal{G} \in \mathcal{D}\}$.*

The proposed problem formulation surfaces two key challenges:

1. *How do we extract the concepts?* Concept extraction poses a significant challenge due to the exponential size of the candidate space. In the worst case, a graph with $n$ nodes can have $2^n$ possible subgraphs. Furthermore, if there are $n$ subgraphs in the dataset, identifying the *unique* subgraphs requires performing $\mathcal{O}(n^2)$ graph isomorphism tests. The computation cost of graph isomorphism grows exponentially with graph size.
2. *How do we uncover the boolean logic over the concepts?* The number of boolean formulas associated with a set of symbols (concepts) increases exponentially with the size of the set. Hence, the scalability challenge is further exacerbated.

## 3 GRAPHTRAIL: Proposed Global Explainer

Fig. 1 presents the pipeline of GRAPHTRAIL. GRAPHTRAIL builds on the observation that any message passing GNN decomposes a graph of $n$ nodes into $n$ *computation trees*. Consequently, only the subgraphs corresponding to these computation trees are processed by the GNN and the rest of the subgraphs are irrelevant to the GNN's predictions. This property allows us to reduce the candidate space size from exponential to linear. Subsequently, the impact of each of the computation trees is assessed through its *Shapley Value* [43], and the top-$k$ trees are sent to the logic formulator. The Boolean logic is revealed by *the symbolic regression* over these computation trees. Symbolic regression aims to discover concise closed-form mathematical equations that best fit a given set of data [24, 31]. In our context, we constrain the regressor over computation trees with boolean operators and fit to the predictions of GNN. We next elaborate on each of these steps.

### 3.1 Computation Framework of Message-passing GNNs

GNNs aggregate messages layer by layer. If $\mathbf{x}_v \in \mathbb{R}^{|F|}$ is the input feature vector for node $v \in \mathcal{V}$, then the $0^{th}$ layer representation of node $v$ is set to $\mathbf{h}_v^0 = \mathbf{x}_v$. In each of the subsequent layers $\ell$,

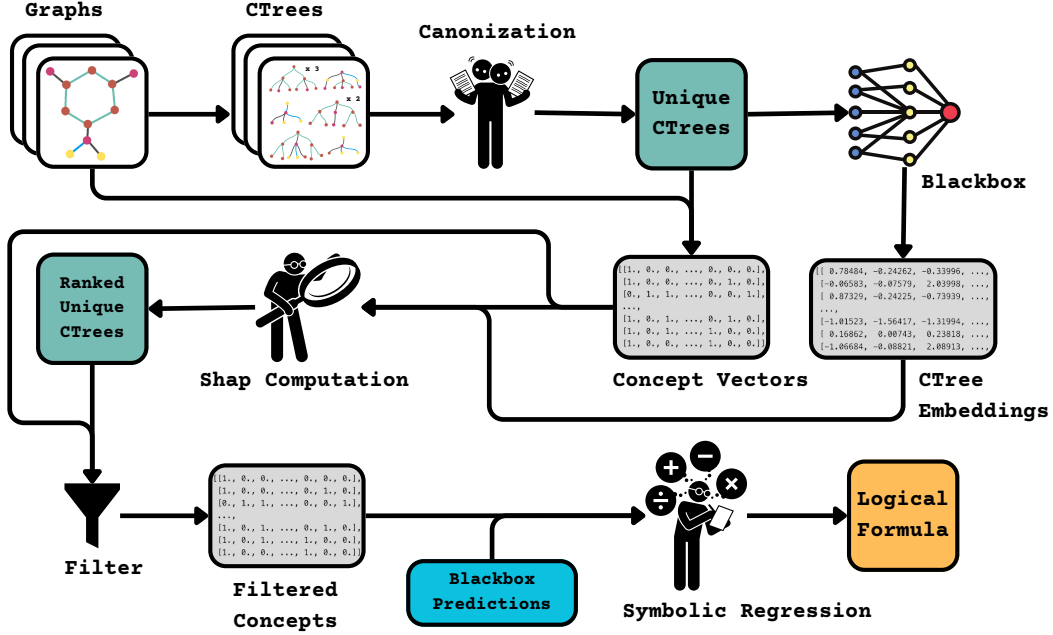

Figure 1: Pipeline of the GRAPHTRAIL algorithm.

GNNs draw messages from their neighbors $\mathcal{N}_v = \{u \mid (u, v \in \mathcal{E})\}$ and aggregate as follows:

$$\mathbf{m}_v^\ell(u) = \text{MSG}^\ell(\mathbf{h}_u^{\ell-1}, \mathbf{h}_v^{\ell-1}) \tag{1}$$

$$\overline{\mathbf{m}}_v^\ell = \text{AGGREGATE}^\ell(\{\!\{\mathbf{m}_v^\ell(u), \forall u \in \mathcal{N}_v\}\!\}) \tag{2}$$

$$\mathbf{h}_v^\ell = \text{MLP}^\ell(\mathbf{h}_v^{\ell-1}, \overline{\mathbf{m}}_v^\ell) \tag{3}$$

Here, $\text{MSG}^\ell$ and $\text{AGGREGATE}^\ell$ can be either pre-defined functions (e.g., MEANPOOL) or neural networks (e.g., GAT [46]). $\{\!\{\cdot\}\!\}$ denotes a multi-set since the same message may be received from multiple nodes. The $\ell^{th}$-layer representation of node $v$ is the embedding generated by passing the aggregated messages through an MLP. Finally, the representation of the entire graph is computed as:

$$\mathbf{h}_\mathcal{G} = \text{POOL}\left(\{\mathbf{h}_v^L \mid \forall v \in \mathcal{V}\}\right) \tag{4}$$

Here, POOL could represent aggregation functions such as MEANPOOL, SUMPOOL, etc., and $L$ is the total number of layers in the GNN. Following the computation of the graph embedding, it is mapped to a distribution $\Phi(\mathcal{G})$ over all class labels by passing $\mathbf{h}_\mathcal{G}$ through an MLP.

### 3.2 Mapping Concepts to Rooted Computation Trees

The message passing architecture of GNNs limits the embedding computation of a node $v$ within the *computation tree* rooted at $v$.

**Definition 5** (Rooted Computation Tree $\mathcal{T}_v^L$). *A computation tree, also known as a receptive field, is a fundamental concept in GNNs that describes how information propagates through the graph during the neural network's operation. Formally, given a graph $\mathcal{G} = (\mathcal{V}, \mathcal{E}, \boldsymbol{X})$, a node $v \in \mathcal{V}$, and the number of layers $L$ in the GNN, the computation tree $\mathcal{T}_v^L$ rooted at $v$ is defined as follows:*

- *Enumerate all paths (including those with repeated vertices) starting from $v$ and extending up to $L$ hops away.*
- *Merge these paths into a tree structure where: i) the root is always $v$, and ii) two nodes from different paths are merged if: a) they are at the same depth in their respective paths, and b) all their ancestors in the paths have already been merged. This tree represents how information flows to node $v$ during $L$ rounds of message passing in the GNN.*

Fig. 2 illustrates the process of constructing the computation trees. We now highlight their properties that enable us to compress and process the concept space in a tractable manner.

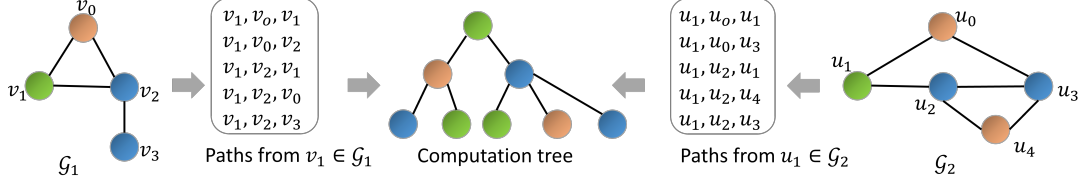

Figure 2: The figure illustrates the process of constructing the computation trees of nodes $v_1 \in \mathcal{G}_1$ and $u_1 \in \mathcal{G}_2$ for $L = 2$. The colors of the nodes represent the node labels. Note that although $v_1$ and $u_1$ are embedded in non-isomorphic $L$-hop neighborhoods, their computation trees are isomorphic.

**Lemma 1.** *In an $L$-layered* GNN*, the computation tree $\mathcal{T}_v^L$ is sufficient to compute node embedding* $\mathbf{h}_v^L, \forall v \in \mathcal{V}$.

*Proof.* In a single layer of message-passing, a node $v$ receives messages from each of its immediate neighbors. Over $L$ layers, $v$ collects messages from nodes within $L$ hops. The computation tree $\mathcal{T}_v^L$ contains all such paths of length up to $L$ and hence is sufficient to compute $\mathbf{h}_v^L$. $\qquad\square$

**Lemma 2.** $\mathcal{T}_v^L \cong \mathcal{T}_u^L \implies \mathbf{h}_v^L = \mathbf{h}_u^L$.

*Proof.* The expressive power of a message-passing GNN is upper bounded by the *Weisfeiler-Lehman test (1-WL)* [53]. This implies that if the $L$-hop neighborhoods of two nodes are indistinguishable by 1-WL, then their representations will be the same. The 1-WL test cannot differentiate between nodes with identical computation trees [44]. $\qquad\square$

Owing to Lemma 1 and Lemma 2, given a graph $\mathcal{G} = (\mathcal{V}, \mathcal{E}, \boldsymbol{X})$, we can decompose it into a multiset of computation trees $\mathcal{T}_\mathcal{G} = \{\!\!\{\mathcal{T}_v^L \mid v \in \mathcal{V}\}\!\!\}$, compute node embeddings $\mathbf{h}_v^L, \forall v \in \mathcal{V}$, and then aggregate them into graph embedding $\mathbf{h}_\mathcal{G}$ (Eq. 4) without incurring any loss of information. This computation structure enables us to shrink the candidate space of concepts, originally defined in Prob. 1, to the set of unique computation trees, i.e., $\mathcal{C} = \bigcup_{\mathcal{G} \in \mathcal{D}} \mathcal{T}_\mathcal{G}$. The reformulation of the concept space imparts several desirable side-effects.

- First, the size of the candidate space of concepts reduces from exponential to $\sum_{\forall \mathcal{G} \in \mathcal{D}} |\mathcal{V}|$, which is linear to the graph size (i.e., number of nodes) and the number of graphs in the dataset.
- Second, while graph isomorphism has a computational cost exponential to the graph size, rooted-tree isomorphism can be performed in time linear to the number of edges in the tree (See App. A for details). Therefore, distilling all computation trees to only the unique ones can be done in linear time.
- Third, GNNs may map non-isomorphic $L$-hop node neighborhoods (subgraphs) to isomorphic computation trees (See Fig. 2). This further reduces the number of concept candidates and, consequently, the computational burden.

### 3.3 Mining Concepts

To mine concepts from the candidate space, we assess their *Shapley values*. Shapley value [43] is a co-operative game theoretic technique where players form coalitions and receive payouts based on their contribution to the outcome. Mathematically, the Shapley value $\psi_i(\Phi)$ of player $i$ is its average marginal contribution across all possible combinations of features.

$$\psi_i(V) = \sum_{S \subseteq \{1, \cdots p\} \setminus \{i\}} \frac{|S|! \, (p - |S| - 1)!}{p!} (V(S \cup \{i\}) - V(S)) \qquad (5)$$

Here, $p$ is the total number of players, $S$ represents a subset of players excluding player $i$ and the *value function* $V(.)$ quantifies the outcome.

With the hypotheses that some combination of computation trees is responsible for GNN's predictions, we aim to leverage Shapley values in identifying this combination. Consequently, the players correspond to computation trees, and the value function corresponds to the performance of the GNN when restricted to the chosen subset of computation trees. We set the value function as the *negated cross-entropy error* of GNN $\Phi$'s prediction (one may use other accuracy measures as well). Thus, Eq. 5 is re-expressed as:

$$\psi_i(\Phi) = \sum_{\forall S, S \subseteq \mathcal{C} \setminus \{C_i\}} \frac{|S|! \, (|\mathcal{C}| - |S| - 1)!}{|\mathcal{C}|!} \left( V(S \cup \{C_i\}) - V(S) \right) \tag{6}$$

$$\text{where, } V(S) = \sum_{\forall \mathcal{G} \in \mathcal{D}} \sum_{j=1}^{c} \mathcal{Y}_j^{\mathcal{G}} \log \left( \Phi \left( \mathcal{G}^S \right)_j \right) \tag{7}$$

In Eq. 6, $\mathcal{C} = \{C_1, \cdots, C_m\}$ is the set of unique computation trees, and $C_i$ denotes the $i$-th one. Eq. 7 represents the negated cross-entropy error. In Eq. 7, we slightly overload the notation introduced in Def. 3 as follows: we assume that the class label information is maintained as a $c$-dimensional one-hot vector. Thus, $\mathcal{Y}_j^{\mathcal{G}}$ is 1 if the true class label of $\mathcal{G}$ is $j$-th label. The GNN $\Phi$ returns a distribution over the class labels, where $\Phi(\mathcal{G}^S)_j$ is the probability of the $j$-th class, when $\mathcal{G}$ is embedded by only considering the computation trees within $S$. Mathematically, this means modifying Eq. 4 as follows:

$$\mathbf{h}_{\mathcal{G}}^S = \text{POOL} \left( \left\{ \mathbf{h}_v^L \mid \forall v \in \mathcal{V} \text{ and } \mathcal{T}_v^L \in S \right\} \right) \tag{8}$$

Finally, we select the top-$k$ computation trees with the highest Shapley value as concepts. We discuss our procedure for selecting $k$ in the next section (§ 3.4).

### 3.3.1 Computational Tractability

While we are able to map the problem of concept mining to Shapley Value computations, several computational challenges arise. Firstly, Shapley value computation necessitates sampling all possible subsets of players (computation trees), leading to exponential computation complexity. Secondly, we need to compute $\mathbf{h}_{\mathcal{G}}^S$, i.e., the graph embedding corresponding to each subset of trees $S$. The naïve approach involves: **(1)** enumerating computation trees rooted in each node, which consumes $\mathcal{O}(|\mathcal{E}|)$ time, **(2)** performing $\mathcal{O}(|S||\mathcal{V}|)$ tree isomorphisms to detect computation trees contained within $S$ and then **(3)** computing Eq. 8. Given that Shapley computation requires drawing multiple samples of $S$ (Eq. 6), the above pipeline for each sample $S$ is prohibitively slow.

Fortunately, there are works to circumvent the first challenge via sampling a small set of coalitions with probabilistic approximations [59, 33]. We use the sampling strategy outlined in [28] (see App. B for details).

The second challenge is unique since computing Shapley on computation trees of GNNs has not been studied. GraphShap [36], the only other work to study Shapley value in the context of explaining graph classification, looks at the specific class of identity-aware graphs, where nodes with unique identities recur across multiple graph views. More importantly, GraphShap is not customized for GNNs and hence the context of computation tree does not arise.

To address the bottleneck of computing Eq. 8, we project each graph into a *concept vector*.

**Definition 6** (Concept vector). *Let $\mathcal{C} = \{C_1, \cdots, C_m\}$ be the set of unique computation trees. The concept vector $\mathbf{C}_{\mathcal{G}} \in \mathbb{Z}_{\geq 0}^m$ of graph $\mathcal{G} = (\mathcal{V}, \mathcal{E}, \mathbf{X})$ is an $m$-dimensional vector where the $i$-th dimension represents the number of computation trees in $\mathcal{G}$ that are isomorphic to the $i$-th concept candidate $C_i \in \mathcal{C}$. Mathematically, $\mathbf{C}_{\mathcal{G}}[i] = |\{v \in \mathcal{V} \mid \mathcal{T}_v^L \cong C_i\}|$.*

The "concept encoding" box in Fig. 1 illustrates this operation. From Lemma 1, the embedding of the root node of $C_i$ depends solely on $C_i$. Hence, we pre-compute and map each $C_i \in \mathcal{C}$ into an embedding $\mathbf{h}_i$. By combining Lemma 2 with Eq.4, we deduce that for both MEANPOOL and SUMPOOL, the two most common aggregators can be computed in $O(|S|)$ time. Specifically,

$$\text{For SUMPOOL,} \quad \mathbf{h}_{\mathcal{G}}^S = \sum_{C_i \in S} \mathbf{C}_{\mathcal{G}}[i] \times \mathbf{h}_i \tag{9}$$

$$\text{For MEANPOOL,} \quad \mathbf{h}_{\mathcal{G}}^S = \frac{1}{|\mathcal{V}|} \sum_{C_i \in S} \mathbf{C}_{\mathcal{G}}[i] \times \mathbf{h}_i \tag{10}$$

Owing to Lemma 1 and Lemma 2, computing the graph embedding simply involves $|S|$ look-ups of vectors instead of fresh rounds of message-passing.

### 3.4 Generation of Logical Rules through Symbolic Regression

Let us denote the top-$k$ computation trees with the $k$ highest Shapley values as $\mathcal{C}^*$. To discover logical rules over $\mathcal{C}^*$, we perform *symbolic regression*. Symbolic regression is a branch of regression analysis that aims to discover symbolic expressions from experimental data [31]. Formally, given a set of $n$

input-output pairs $\{(\mathbf{x}_i, y_i)\}_{i=1}^n$, where $\mathbf{x}_i \in \mathbb{R}^d$ is an input vector, and $y_i \in \mathbb{R}$ is the output value (or label), and a set of operators, such as addition, subtraction, logarithm, trigonometric functions, boolean logic, etc., the goal is to find a symbolic equation $e$ and corresponding function $f_e$ such that $\forall i, \ y_i \approx f_e(\mathbf{x}_i)$.

To perform symbolic regression over computation trees, we first project the concept vector of each graph $\mathcal{G} = (\mathcal{V}, \mathcal{E}, \boldsymbol{X})$ on the top-$k$ trees $\mathcal{C}^*$. This distilled concept vector is denoted as $\boldsymbol{C}_{\mathcal{G}}^* \in \mathbb{Z}_{\geq 0}^k$, where $\boldsymbol{C}_{\mathcal{G}}^*[i] = |\{v \in \mathcal{V} \mid \mathcal{T}_v^L \cong C_i\}|$, with $C_i$ representing the $i^{\text{th}}$ ranked computation tree in $\mathcal{C}^*$ based on Shapley value. While symbolic regression can accommodate a wide array of operators, mathematical operations such as addition or multiplication are not meaningful when applied to computation trees. Hence, to facilitate human interpretability, we restrict to the boolean operators: conjunction ($\wedge$), disjunction ($\vee$), negation ($\neg$), and XOR ($\oplus$). We aim to learn a symbolic function $f_c$ for each class label $c$ predicted by the GNN. Let $\mathcal{D}_c = \{\mathcal{G} \in \mathcal{D} \mid \Phi(\mathcal{G})^* = c\}$ be the subset of graphs where the predicted label is $c$. The symbolic function $f_c$ is identified by minimizing a multi-objective loss function that balances prediction error and model complexity. Formally,

$$\mathcal{L}\left(f_c\right) = Error(f_c) + (\alpha)Complexity(f_c) \tag{11}$$

$Complexity(f_c)$ corresponds to the number of boolean operators and variables in $f_c$. $\alpha$ is a weighting hyper-parameter. Error corresponds to the number of disagreements with the GNN. Formally,

$$Error\left(f_c\right) = \sum_{\forall \mathcal{G} \in \mathcal{D}_c} 1 - f_c(\mathcal{G}) + \sum_{\forall \mathcal{G}' \in \mathcal{D} \setminus \mathcal{D}_c} f_c(\mathcal{G}) \tag{12}$$

Since $f_c$ is a boolean function, it returns either `TRUE` (equivalently 1) or `FALSE` (0).

Symbolic regression poses a combinatorial optimization challenge, as the number of potential functions increases exponentially with the number of symbols, which is $k$ in our case. Fortunately, the area is rich with several studies [31]. We use [8], which leverages a multi-population evolutionary algorithm to efficiently optimize and identify symbolic expressions (See App. C for details).

As depicted in Fig. 1, following the generation of boolean functions, GRAPHTRAIL concludes.

**Identifying $k$ – the number of concepts:** The higher the value of $k$, the more expressive symbolic regression is to fit to the prediction data by GNN. On the other hand, the computation cost grows monotonically with $k$. To optimize this balance, we start with a small value of $k$, and incrementally increase it until the accuracy plateaus, similar to using the patience parameter to determine the number of epochs in machine learning model training.

## 4 Experiments

In this section, we evaluate GRAPHTRAIL and benchmark its performance in translating predictions of various GNN architectures across diverse datasets and pooling layers. More experiments can be found in the appendix. The codebase of GRAPHTRAIL is shared at https://github.com/idea-iitd/GraphTrail.

### 4.1 Experimental Setup

**Baselines:** As discussed in § 1, GLGEXPLAINER [2] is the only existing algorithm that fits a logical formula to GNN predictions, making it our primary baseline. Recall, GLGEXPLAINER generates a formula over vectors representing embeddings of subgraph clusters rather than individual subgraphs, rendering it non-interpretable. The formula is applied to an input graph by first processing it through an instance-level explainer, embedding the explanation subgraphs into feature space, and assigning them to the closest cluster. These cluster vectors are then considered present in the input graph. To make the formula interpretable, each cluster vector is replaced by the subgraph closest to the cluster representation in the embedding space. A subgraph (concept) $\mathcal{G}_S$ in the formula is considered present in an input graph $\mathcal{G}$ if $\mathcal{G}_S \subseteq \mathcal{G}$. We term this version GLGEXPLAINER-iso.

We use the authors' implementation of GLGEXPLAINER with the recommended parameters and employ PGEXPLAINER [29] as the instance-level explainer, as suggested. The implementation of PGEXPLAINER is available in Pytorch Geometric.

**GRAPHTRAIL-S:** Shapley value computation brings the computational bottleneck in the pipeline, which involves calculating the global Shapley value for each computation tree on each graph. This process can become impractical for datasets with a large number of graphs. To address this issue, we propose GRAPHTRAIL-S, an efficient version of GRAPHTRAIL. Note that we are interested in the ranking according to the Shapley values and not the values themselves. If a subset of the

| | BAMultiShapes | MUTAG | Mutagenicity | NCI1 |
|---|---|---|---|---|
| GLG | $0.48 \pm 0.02$ | $0.74 \pm 0.10$ | $0.62 \pm 0.03$ | $0.57 \pm 0.04$ |
| GLG-iso | $0.00 \pm 0.00$ | $0.61 \pm 0.20$ | $0.53 \pm 0.06$ | $0.49 \pm 0.05$ |
| G-TRAIL-S | $\underline{0.86 \pm 0.01}$ | $\underline{0.78 \pm 0.08}$ | $\mathbf{0.72 \pm 0.01}$ | $\underline{0.72 \pm 0.02}$ |
| G-TRAIL | $\mathbf{0.87 \pm 0.02}$ | $\mathbf{0.82 \pm 0.09}$ | $\mathbf{0.72 \pm 0.01}$ | $\mathbf{0.72 \pm 0.02}$ |

Table 1: Average fidelity of the formulae across three seeds. The best and second best results are shown in bold and underlined, respectively. G-TRAIL and GLG represents GRAPHTRAIL and GLGEXPLAINER respectively.

| | GNN Architecture | | | | | | POOL | | | | | |
|---|---|---|---|---|---|---|---|---|---|---|---|---|
| | GCN | | GAT | | GIN | | SUM | | MEAN | | MAX | |
| | G-TRAIL | GLG | G-TRAIL | GLG | G-TRAIL | GLG | G-TRAIL | GLG | G-TRAIL | GLG | G-TRAIL | GLG |
| BAMultiShapes | **0.81** | 0.49 | **1.00** | 0.93 | **0.87** | 0.48 | **0.87** | 0.48 | **0.88** | 0.50 | **0.89** | 0.52 |
| MUTAG | **0.87** | 0.80 | **0.82** | 0.74 | **0.87** | 0.55 | **0.82** | 0.74 | **0.77** | 0.54 | **0.78** | 0.74 |
| Mutagenicity | **0.72** | 0.63 | **0.72** | 0.62 | **0.73** | 0.64 | **0.72** | 0.62 | **0.70** | 0.62 | **0.69** | 0.68 |
| NCI1 | **0.75** | 0.59 | **0.73** | 0.65 | **0.72** | 0.57 | **0.72** | 0.57 | **0.70** | 0.59 | **0.66** | 0.62 |

Table 2: Comparison of Fidelity across GNN architectures and POOL layers. The best result in each dataset and category is highlighted in bold.

dataset can emulate the entire dataset, the generated ranking is expected to remain the same. Thus, GRAPHTRAIL-S creates a stratified split of the training dataset that maintains the class ratios and computes the global Shapley values using this subset. The rest of the pipeline remains unchanged, allowing for a more efficient yet effective method. This is also shown in the results from Table 1 where GRAPHTRAIL-s is slightly inferior to GRAPHTRAIL.

**Datasets:** We use four benchmark datasets listed in Table C (App. D). While NCI1 [49], MUTAG [17], and Mutagenicity [39, 20] are collections of molecules, BAMultiShapes [55] is a synthetic dataset specifically curated to benchmark global GNN explainers and associated with ground truth logical explanations. Further details are presented in App. D.

**Evaluation framework:** While we benchmark against various GNN architectures and POOL layers (Eq. 4), the default architecture is set to GAT for MUTAG and Mutagenicity and GIN for the other two. We use SUMPOOL as the default across datasets. The accuracies of the GNN architectures across all POOL choices are presented in Table D in the Appendix. Details on the hardware platform, train-val-test splits and other parameters are provided in App. E. All experiments have been repeated with three seeds and we report the means and standard deviations.

**Metrics:** We evaluate explainer faithfulness using *Fidelity*; the ratio of graphs where the logical formula matches the GNN class label. Additional metrics can be found in the appendix.

## 4.2 Fidelity

Is GRAPHTRAIL faithful in translating GNN through logical rules? Table 1 provides the answer with several key observations.

First, both GRAPHTRAIL and GRAPHTRAIL-s significantly outperform GLGEXPLAINER. GRAPH-TRAIL computes Shapley values by evaluating all subsets of concepts, while GRAPHTRAIL-s employs a faster, sampling-based method. The key reason for the superior performance of GRAPHTRAIL and GRAPHTRAIL-s over GLGEXPLAINER lies in aligning the concept definitions with the fundamental computational units of a GNN, namely computational trees. GRAPHTRAIL then mines these concepts with a global perspective by analyzing their Shapley values. In contrast, GLGEXPLAINER depends on instance-level explainers, which lack the ability to detect global patterns.

Second, GLGEXPLAINER-iso consistently underperforms compared to GLGEXPLAINER. This is because the clusters of instance-level subgraph explanations in GLGEXPLAINER-iso are not *pure* (See App. I). Although they are grouped based on embedding distance, their structures vary significantly. Representing a cluster by an exemplar subgraph fails to capture the diversity effectively.

Finally, GLGEXPLAINER results do not match the original results reported in [2]. In App. H, we discuss the causes, including the identification of test data leakage in the Mutagenicity data set.

## 4.3 Robustness

*Varying architectures and pooling mechanisms.* In Table 2, we present the robustness of GRAPHTRAIL and GLGEXPLAINER across various GNN architectures and POOL layers. GRAPHTRAIL achieves

higher fidelity than GLGEXPLAINER across all scenarios. This result further substantiates the superior ability of GRAPHTRAIL in faithfully translating GNN outcomes.

*Varying $k$.* Fig. 3 explores the relationship between the number of mined concepts ($k$) and the fidelity achieved by GRAPHTRAIL across various datasets. Recall that higher values of $k$ allow for greater expressivity in symbolic regression. Consistent with this, the results exhibit an increasing trend, with fidelity approaching saturation at $k = 50$ for most datasets.

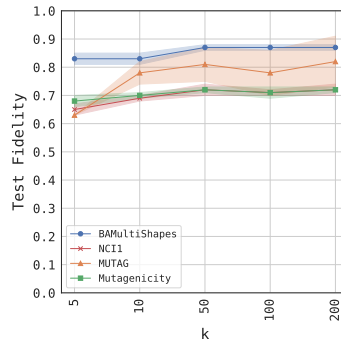

### 4.4 Data Efficiency

In Fig. 4, we assess the explainers' effectiveness in low-data regimes by varying the train data volume on the $x$-axis and measuring Fidelity on the $y$-axis. We expect Fidelity to decrease with lesser data. In GRAPHTRAIL, the decrease in Fidelity with lesser data is minor, indicating robust efficacy even with limited data and that it can be used in low-resource settings. Notably, the higher average Fidelity and lower standard deviation in GRAPHTRAIL across sizes indicate enhanced stability.

Figure 3: Impact of $k$ (number of concepts) on Fidelity.

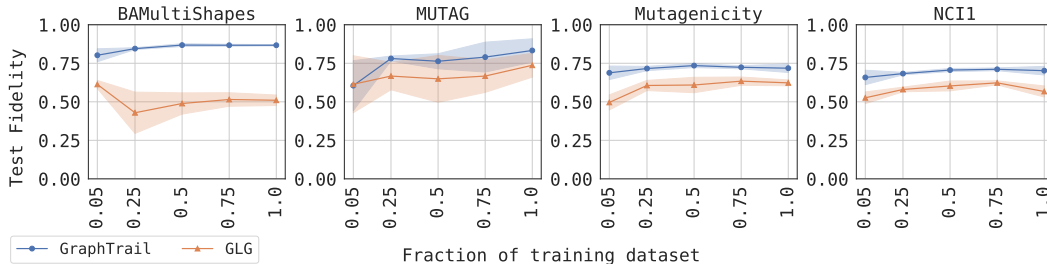

Figure 4: Results on the test set fidelity averaged over three seeds while varying the training data size.

### 4.5 Visual Analysis of Rules

Fig. 5 presents the rules inferred by GRAPHTRAIL and GLGEXPLAINER for the datasets MUTAG, Mutagenicity, and BAMultiShapes.

**MUTAG**: According to [9], compounds with rings and electron-attracting group elements conjugated with nitro groups enhance mutagenicity. GRAPHTRAIL identifies both the ring structures as well as the nitro group as identifiers of mutagenicity in this dataset. In contrast, GLGEXPLAINER fails to identify the ring structure.

**Mutagenicity:** Prior work [55, 45, 29, 2] often limits the mutagenic behavior in this dataset to $NO_2$ and $NH_2$, i.e., the nitro and amine groups. However, in [20], eight general toxicophores are capable of identifying 75% of all mutagens in the dataset (see Fig. F [20]). As seen in Fig. 5, without any supervision, GRAPHTRAIL identifies and incorporates six of them in its formula, viz. aromatic nitro, aromatic amine, nitroso, unsubstituted heteroatom-bonded heteroatom, azo-type, and polycyclic aromatic systems. The azo-type and the unsubstituted heteroatom-bonded heteroatom groups have not been identified explicitly but are potential extensions of two of the identified structures.

**BAMultiShapes**: BAMultiShapes has known ground truth logical rules for its class labels. While GRAPHTRAIL does not perfectly recover these rules, GRAPHTRAIL's inferred concepts and their logical combination exhibit greater similarity to the ground truth compared to GLGEXPLAINER's results. It should be noted that explainers try to find what the model has learned rather than the ground truth. One striking observation is that GLGEXPLAINER's extracted concepts bear no similarity to the ground truth and two of the prototypes have no graphs in their clusters and are therefore empty. In contrast, GRAPHTRAIL identifies concepts that are isomorphic to the grid and house motifs present in the ground truth.

## 5 Conclusions, Limitations and Future Directions

In this work, we have designed an end-to-end, post-hoc, global graph neural network (GNN) explainer called GRAPHTRAIL. We have formulated the problem of translating a message-passing GNN model

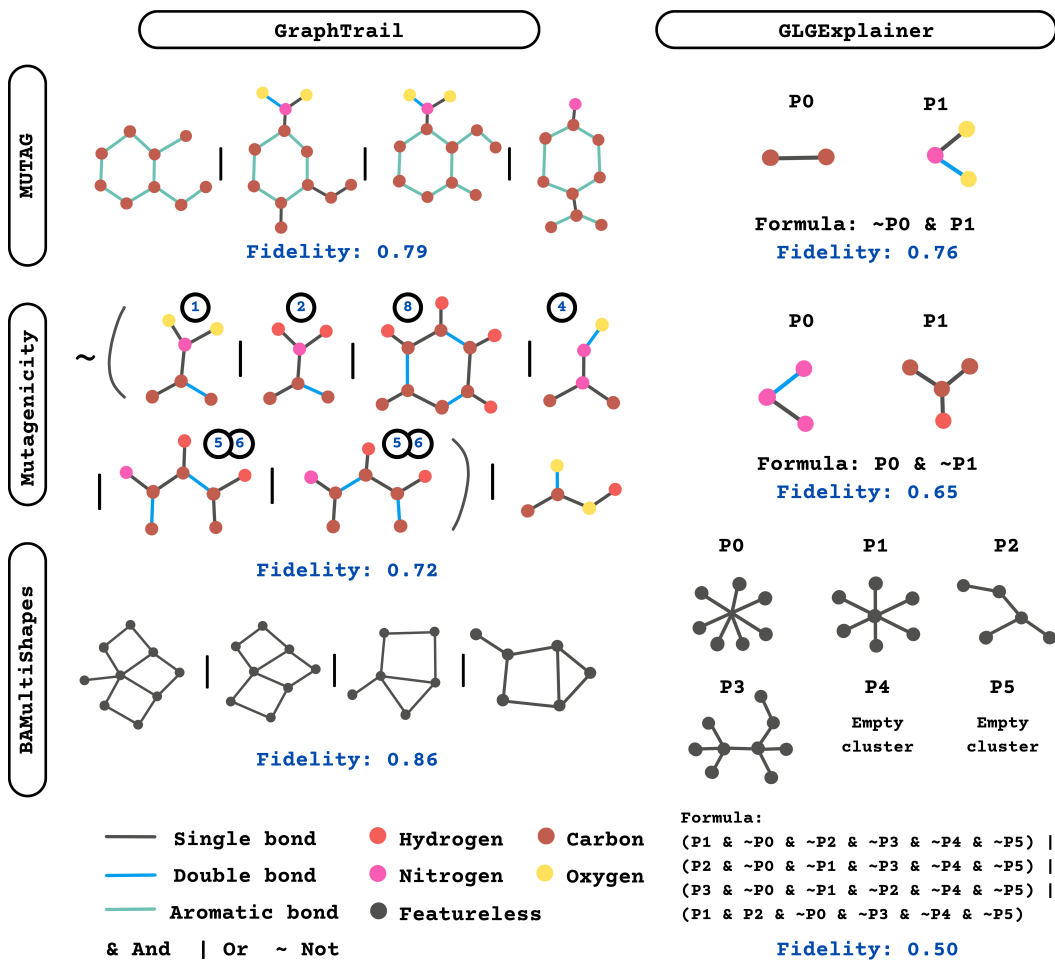

Figure 5: Visual inspection of the formulae generated by GRAPHTRAIL and GLGEXPLAINER. The encircled numbers in Mutagenicity are toxicophore IDs from Fig. F (in Appendix). The structures with two IDs in Mutagenicity can be extended to both toxicophores.

for graph classification into a human-interpretable logic formula over subgraph concepts without assuming the concepts are pre-generated through a separate algorithm. GRAPHTRAIL leverages several novel insights such as the decomposition of graphs into computation trees by message-passing GNNs, which reduces the search space for concepts from exponential number of all possible subgraphs to a linear space of computation trees. GRAPHTRAIL evaluates the impact of these computation trees using Shapley values and then creates a boolean formula via symbolic regression only with the important ones. Extensive empirical analyses across diverse datasets and different GNN architectures demonstrate GRAPHTRAIL's effectiveness in capturing the underlying logical relationships within complex GNN models for a wide range of scenarios in a human-interpretable form.

**Limitations:** While both GRAPHTRAIL and GLGEXPLAINER express rules as logical combinations of concept's presence or absence, they currently lack the ability to capture the multiplicity of a concept's occurrence within a graph. Finally, we believe attempting to interpret GNNs with subgraph level concepts creates a dilemma between interpretability and faithfulness to the computational mechanism of GNNs. To elaborate, in GRAPHTRAIL, since symbolic regression generates the formula over computation trees, we map it back to the graph space, by replacing each computation tree with the most frequent $L$-hop ego graph that generates this tree. Note, GNNs operate at the level of computation trees. Given any formula at a subgraph level, a new formula can be generated with identical fidelity, by simply replacing each subgraph concept with one that produces the same set of computation trees (recall Fig. 2). However, when concepts are lowered to computation tree level, human interpetability is compromised. It is worth exploring how the best balance between human interpretability [13] and staying faithful to GNN computation structure can be obtained.

## 6 Acknowledgements

Burouj Armgaan acknowledges the support of the Prime Minister's Research Fellowship (PMRF) for funding this research.

## Footnotes

[1]Trail also implies a pathway of clues, which aligns with our problem

## References

[1] Carlo Abrate and Francesco Bonchi. Counterfactual graphs for explainable classification of brain networks. In *KDD*, page 2495–2504, 2021. (Cited on p. **1** ↩)

[2] Steve Azzolin, Antonio Longa, Pietro Barbiero, Pietro Lio, and Andrea Passerini. Global explainability of GNNs via logic combination of learned concepts. In *The Eleventh International Conference on Learning Representations*, 2023. (Cited on pp. **2, 3, 7, 8, 9, 15, and 19** ↩)

[3] Mohit Bajaj, Lingyang Chu, Zi Yu Xue, Jian Pei, Lanjun Wang, Peter Cho-Ho Lam, and Yong Zhang. Robust counterfactual explanations on graph neural networks. In A. Beygelzimer, Y. Dauphin, P. Liang, and J. Wortman Vaughan, editors, *Advances in Neural Information Processing Systems*, 2021. (Cited on pp. **1 and 15** ↩)

[4] Federico Baldassarre and Hossein Azizpour. Explainability techniques for graph convolutional networks. *arXiv preprint arXiv:1905.13686*, 2019. (Cited on p. **15** ↩)

[5] Vaibhav Bihani, Sahil Manchanda, Srikanth Sastry, Sayan Ranu, and NM Krishnan. Stridernet: A graph reinforcement learning approach to optimize atomic structures on rough energy landscapes. In *ICML*, 2023. (Cited on p. **1** ↩)

[6] Chirag Chhablani, Sarthak Jain, Akshay Channesh, Ian A Kash, and Sourav Medya. Game-theoretic counterfactual explanation for graph neural networks. In *Proceedings of the ACM on Web Conference 2024*, pages 503–514, 2024. (Cited on p. **1** ↩)

[7] Yun Chi, Yirong Yang, and Richard Muntz. Canonical forms for labelled trees and their applications in frequent subtree mining. *Knowl. Inf. Syst.*, 8:203–234, 08 2005. (Cited on p. **15** ↩)

[8] Miles Cranmer. Interpretable machine learning for science with pysr and symbolicregression. jl. *arXiv preprint arXiv:2305.01582*, 2023. (Cited on pp. **7 and 17** ↩)

[9] Asim Kumar Debnath, Rosa L Lopez de Compadre, Gargi Debnath, Alan J Shusterman, and Corwin Hansch. Structure-activity relationship of mutagenic aromatic and heteroaromatic nitro compounds. correlation with molecular orbital energies and hydrophobicity. *Journal of medicinal chemistry*, 34(2):786–797, 1991. (Cited on pp. **9 and 15** ↩)

[10] Dobrik Georgiev, Pietro Barbiero, Dmitry Kazhdan, Petar Veličković, and Pietro Liò. Algorithmic concept-based explainable reasoning. In *Proceedings of the AAAI Conference on Artificial Intelligence*, volume 36, pages 6685–6693, 2022. (Cited on p. **2** ↩)

[11] Mridul Gupta, Hariprasad Kodamana, and Sayan Ranu. FRIGATE: Frugal spatio-temporal forecasting on road networks. In *29th SIGKDD Conference on Knowledge Discovery and Data Mining*, 2023. (Cited on p. **1** ↩)

[12] Shubham Gupta, Sahil Manchanda, Srikanta Bedathur, and Sayan Ranu. Tigger: Scalable generative modelling for temporal interaction graphs. In *Proceedings of the AAAI Conference on Artificial Intelligence*, volume 36, pages 6819–6828, 2022. (Cited on p. **1** ↩)

[13] Pantea Habibi, Peyman Bagershahi, Sourav Medya, and Debaleena Chattopadhyay. Design requirements for human-centered graph neural network explanations. *arXiv preprint arXiv:2405.06917*, 2024. (Cited on p. **10** ↩)

[14] William L. Hamilton, Rex Ying, and Jure Leskovec. Inductive representation learning on large graphs. In *Proceedings of the 31st International Conference on Neural Information Processing Systems*, NIPS'17, page 1025–1035, Red Hook, NY, USA, 2017. Curran Associates Inc. (Cited on pp. **1 and 3** ↩)

[15] Qiang Huang, Makoto Yamada, Yuan Tian, Dinesh Singh, and Yi Chang. Graphlime: Local interpretable model explanations for graph neural networks. *IEEE Transactions on Knowledge and Data Engineering*, 2022. (Cited on pp. **1 and 15** ↩)

[16] Zexi Huang, Mert Kosan, Sourav Medya, Sayan Ranu, and Ambuj Singh. Global counterfactual explainer for graph neural networks. In *Proceedings of the Sixteenth ACM International Conference on Web Search and Data Mining*, pages 141–149, 2023. (Cited on pp. **2 and 15** ↩)

[17] Sergei Ivanov, Sergei Sviridov, and Evgeny Burnaev. Understanding isomorphism bias in graph data sets. *Geometric Learning and Graph Representations ICLR Workshop*, 2019. (Cited on pp. **8 and 17** ↩)

[18] Jayant Jain, Vrittika Bagadia, Sahil Manchanda, and Sayan Ranu. Neuromlr: Robust & reliable route recommendation on road networks. *Advances in Neural Information Processing Systems*, 34:22070–22082, 2021. (Cited on p. **1** ↩)

[19] Jaykumar Kakkad, Jaspal Jannu, Kartik Sharma, Charu Aggarwal, and Sourav Medya. A survey on explainability of graph neural networks. *arXiv preprint arXiv:2306.01958*, 2023. (Cited on p. **1** ↩)

[20] Jeroen Kazius, Ross McGuire, and Roberta Bursi. Derivation and validation of toxicophores for mutagenicity prediction. *Journal of medicinal chemistry*, 48(1):312–320, 2005. (Cited on pp. **8, 9, and 17** ↩)

[21] Been Kim, Martin Wattenberg, Justin Gilmer, Carrie Cai, James Wexler, Fernanda Viegas, et al. Interpretability beyond feature attribution: Quantitative testing with concept activation vectors (tcav). In *International conference on machine learning*, pages 2668–2677. PMLR, 2018. (Cited on p. **3** ↩)

[22] Thomas N Kipf and Max Welling. Semi-supervised classification with graph convolutional networks. *arXiv preprint arXiv:1609.02907*, 2016. (Cited on pp. **1 and 3** ↩)

[23] Mert Kosan, Samidha Verma, Burouj Armgaan, Khushbu Pahwa, Ambuj Singh, Sourav Medya, and Sayan Ranu. GNNX-BENCH: Unravelling the utility of perturbation-based GNN explainers through in-depth benchmarking. In *The Twelfth International Conference on Learning Representations*, 2024. (Cited on p. **1** ↩)

[24] John R. Koza. *Genetic programming: on the programming of computers by means of natural selection*. MIT Press, Cambridge, MA, USA, 1992. (Cited on p. **3** ↩)

[25] Wanyu Lin, Hao Lan, and Baochun Li. Generative causal explanations for graph neural networks. In *International Conference on Machine Learning*, pages 6666–6679. PMLR, 2021. (Cited on pp. **1 and 15** ↩)

[26] Shengyao Lu, Keith G Mills, Jiao He, Bang Liu, and Di Niu. GOAt: Explaining graph neural networks via graph output attribution. In *The Twelfth International Conference on Learning Representations*, 2024. (Cited on pp. **1 and 15** ↩)

[27] Ana Lucic, Maartje A Ter Hoeve, Gabriele Tolomei, Maarten De Rijke, and Fabrizio Silvestri. Cf-gnnexplainer: Counterfactual explanations for graph neural networks. In *AISTATS*, pages 4499–4511, 2022. (Cited on pp. **1 and 15** ↩)

[28] Scott M Lundberg and Su-In Lee. A unified approach to interpreting model predictions. *Advances in neural information processing systems*, 30, 2017. (Cited on pp. **6 and 16** ↩)

[29] Dongsheng Luo, Wei Cheng, Dongkuan Xu, Wenchao Yu, Bo Zong, Haifeng Chen, and Xiang Zhang. Parameterized explainer for graph neural network. *Advances in neural information processing systems*, 33:19620–19631, 2020. (Cited on pp. **1, 2, 7, 9, 15, and 19** ↩)

[30] Jing Ma, Ruocheng Guo, Saumitra Mishra, Aidong Zhang, and Jundong Li. Clear: Generative counterfactual explanations on graphs. *arXiv preprint arXiv:2210.08443*, 2022. (Cited on p. **15** ↩)

[31] Nour Makke and Sanjay Chawla. Interpretable scientific discovery with symbolic regression: a review. *Artificial Intelligence Review*, 57(1):2, 2024. (Cited on pp. **3, 6, 7, and 17** ↩)

[32] Sahil Manchanda, Akash Mittal, Anuj Dhawan, Sourav Medya, Sayan Ranu, and Ambuj Singh. Gcomb: Learning budget-constrained combinatorial algorithms over billion-sized graphs. *Advances in Neural Information Processing Systems*, 33:20000–20011, 2020. (Cited on p. **1** ↩)

[33] Sourav Medya, Tiyani Ma, Arlei Silva, and Ambuj Singh. A game theoretic approach for core resilience. In *International Joint Conferences on Artificial Intelligence Organization*, 2020. (Cited on p. **6** ↩)

[34] Peter Müller, Lukas Faber, Karolis Martinkus, and Roger Wattenhofer. Graphchef: Learning the recipe of your dataset. In *ICML 3rd Workshop on Interpretable Machine Learning in Healthcare (IMLH)*, 2023. (Cited on p. **2** ↩)

[35] Sunil Nishad, Shubhangi Agarwal, Arnab Bhattacharya, and Sayan Ranu. Graphreach: Position-aware graph neural network using reachability estimations. *IJCAI*, 2021. (Cited on p. **1** ↩)

[36] Alan Perotti, Paolo Bajardi, Francesco Bonchi, and André Panisson. Explaining identity-aware graph classifiers through the language of motifs. In *2023 International Joint Conference on Neural Networks (IJCNN)*, pages 1–8. IEEE, 2023. (Cited on p. **6** ↩)

[37] Phillip E Pope, Soheil Kolouri, Mohammad Rostami, Charles E Martin, and Heiko Hoffmann. Explainability methods for graph convolutional neural networks. In *Proceedings of the IEEE/CVF conference on computer vision and pattern recognition*, pages 10772–10781, 2019. (Cited on p. **15** ↩)

[38] Rishabh Ranjan, Siddharth Grover, Sourav Medya, Venkatesan Chakaravarthy, Yogish Sabharwal, and Sayan Ranu. Greed: A neural framework for learning graph distance functions. In *Advances in Neural Information Processing Systems*, 2022. (Cited on p. **1** ↩)

[39] Kaspar Riesen and Horst Bunke. Iam graph database repository for graph based pattern recognition and machine learning. In *Joint IAPR International Workshops on Statistical Techniques in Pattern Recognition (SPR) and Structural and Syntactic Pattern Recognition (SSPR)*, pages 287–297. Springer, 2008. (Cited on pp. **8 and 17** ↩)

[40] Thomas Schnake, Oliver Eberle, Jonas Lederer, Shinichi Nakajima, Kristof T Schütt, Klaus-Robert Müller, and Grégoire Montavon. Higher-order explanations of graph neural networks via relevant walks. *IEEE transactions on pattern analysis and machine intelligence*, 44(11):7581–7596, 2021. (Cited on p. **15** ↩)

[41] Rishi Shah, Krishnanshu Jain, Sahil Manchanda, Sourav Medya, and Sayan Ranu. Neurocut: A neural approach for robust graph partitioning. In *Proceedings of the 30th ACM SIGKDD Conference on Knowledge Discovery and Data Mining*, KDD '24, page 2584–2595, New York, NY, USA, 2024. Association for Computing Machinery. (Cited on p. **1** ↩)

[42] Caihua Shan, Yifei Shen, Yao Zhang, Xiang Li, and Dongsheng Li. Reinforcement learning enhanced explainer for graph neural networks. In *NeurIPS 2021*, December 2021. (Cited on p. **1** ↩)

[43] L. Shapley. *7. A Value for n-Person Games. Contributions to the Theory of Games II (1953) 307-317.*, pages 69–79. Princeton University Press, Princeton, 1997. (Cited on pp. **3, 5, and 16** ↩)

[44] Nino Shervashidze, Pascal Schweitzer, Erik Jan van Leeuwen, Kurt Mehlhorn, and Karsten M. Borgwardt. Weisfeiler-lehman graph kernels. *J. Mach. Learn. Res.*, 12(null):2539–2561, nov 2011. (Cited on p. **5** ↩)

[45] Juntao Tan, Shijie Geng, Zuohui Fu, Yingqiang Ge, Shuyuan Xu, Yunqi Li, and Yongfeng Zhang. Learning and evaluating graph neural network explanations based on counterfactual and factual reasoning. In *Proceedings of the ACM Web Conference 2022*, WWW '22, page 1018–1027, 2022. (Cited on pp. **1, 9, and 15** ↩)

[46] Petar Veličković, Guillem Cucurull, Arantxa Casanova, Adriana Romero, Pietro Liò, and Yoshua Bengio. Graph attention networks. In *International Conference on Learning Representations*, 2018. (Cited on pp. **1, 3, and 4** ↩)

[47] Samidha Verma, Burouj Armgaan, Sourav Medya, and Sayan Ranu. InduCE: Inductive counterfactual explanations for graph neural networks. *Transactions on Machine Learning Research*, 2024. (Cited on pp. **1 and 15** ↩)

[48] Minh Vu and My T Thai. Pgm-explainer: Probabilistic graphical model explanations for graph neural networks. *Advances in neural information processing systems*, 33:12225–12235, 2020. (Cited on p. **15** ↩)

[49] Nikil Wale and George Karypis. Comparison of descriptor spaces for chemical compound retrieval and classification. In *ICDM*, 2006. (Cited on pp. **8 and 17** ↩)

[50] Xiaoqi Wang and Han Wei Shen. GNNInterpreter: A probabilistic generative model-level explanation for graph neural networks. In *The Eleventh International Conference on Learning Representations*, 2023. (Cited on p. **2** ↩)

[51] Geemi P Wellawatte, Aditi Seshadri, and Andrew D White. Model agnostic generation of counterfactual explanations for molecules. *Chemical science*, 13(13):3697–3705, 2022. (Cited on p. **1** ↩)

[52] Yaochen Xie, Sumeet Katariya, Xianfeng Tang, Edward Huang, Nikhil Rao, Karthik Subbian, and Shuiwang Ji. Task-agnostic graph explanations. *NeurIPS*, 2022. (Cited on p. **15** ↩)

[53] Keyulu Xu, Weihua Hu, Jure Leskovec, and Stefanie Jegelka. How powerful are graph neural networks? In *International Conference on Learning Representations*, 2019. (Cited on pp. **1, 3, and 5** ↩)

[54] Han Xuanyuan, Pietro Barbiero, Dobrik Georgiev, Lucie Charlotte Magister, and Pietro Lió. Global concept-based interpretability for graph neural networks via neuron analysis. In *AAAI*, 2023. (Cited on pp. **2 and 15** ↩)

[55] Zhitao Ying, Dylan Bourgeois, Jiaxuan You, Marinka Zitnik, and Jure Leskovec. Gnnexplainer: Generating explanations for graph neural networks. *Advances in neural information processing systems*, 32, 2019. (Cited on pp. **1, 8, 9, 15, 17, and 19** ↩)

[56] Hao Yuan, Jiliang Tang, Xia Hu, and Shuiwang Ji. Xgnn: Towards model-level explanations of graph neural networks. In *Proceedings of the 26th ACM SIGKDD International Conference on Knowledge Discovery & Data Mining*, pages 430–438, 2020. (Cited on pp. **2 and 15** ↩)

[57] Hao Yuan, Haiyang Yu, Shurui Gui, and Shuiwang Ji. Explainability in graph neural networks: A taxonomic survey. *IEEE Transactions on Pattern Analysis and Machine Intelligence*, 2022. (Cited on p. **1** ↩)

[58] Hao Yuan, Haiyang Yu, Jie Wang, Kang Li, and Shuiwang Ji. On explainability of graph neural networks via subgraph explorations. In *ICML*, pages 12241–12252. PMLR, 2021. (Cited on pp. **1 and 15** ↩)

[59] Jiayao Zhang, Qiheng Sun, Jinfei Liu, Li Xiong, Jian Pei, and Kui Ren. Efficient sampling approaches to shapley value approximation. *Proceedings of the ACM on Management of Data*, 1(1):1–24, 2023. (Cited on p. **6** ↩)

[60] Yue Zhang, David Defazio, and Arti Ramesh. Relex: A model-agnostic relational model explainer. In *Proceedings of the 2021 AAAI/ACM Conference on AI, Ethics, and Society*, pages 1042–1049, 2021. (Cited on p. **15** ↩)

[61] Zaixi Zhang, Qi Liu, Hao Wang, Chengqiang Lu, and Cheekong Lee. Protgnn: Towards self-explaining graph neural networks. *AAAI*, 2021. (Cited on p. **2** ↩)

# Appendix

| Toxicophore name | Aromatic Nitro | Aromatic Amine | Three-membered heterocycle | Nitroso | Unsubstituted heteroatom-bonded heteroatom | Azo-type | Aliphatic Halide | Polycyclic Aromatic System |
|---|---|---|---|---|---|---|---|---|
| Substructure representation | 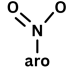 | 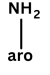 | NH, O, S | 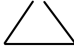 | $NH_2$, OH  N, O | 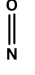 | Cl, Br, I | arom. rings  arom. rings |
| Identifier | 1 | 2 | 3 | 4 | 5 | 6 | 7 | 8 |

Figure F: The eight toxicophores in the Mutagenicity dataset that are capable of identifying 75% of all mutagens in the dataset [9]. "aro" indicates an aromatic atom. "arom. rings" indicates an atom that is part of multiple aromatic rings. The toxicophores identified by GRAPHTRAIL are colored green. Toxicophores 5 and 6 are underlined as they are not explicitly identified but are potential extensions to two of the identified structures in Fig. 5.

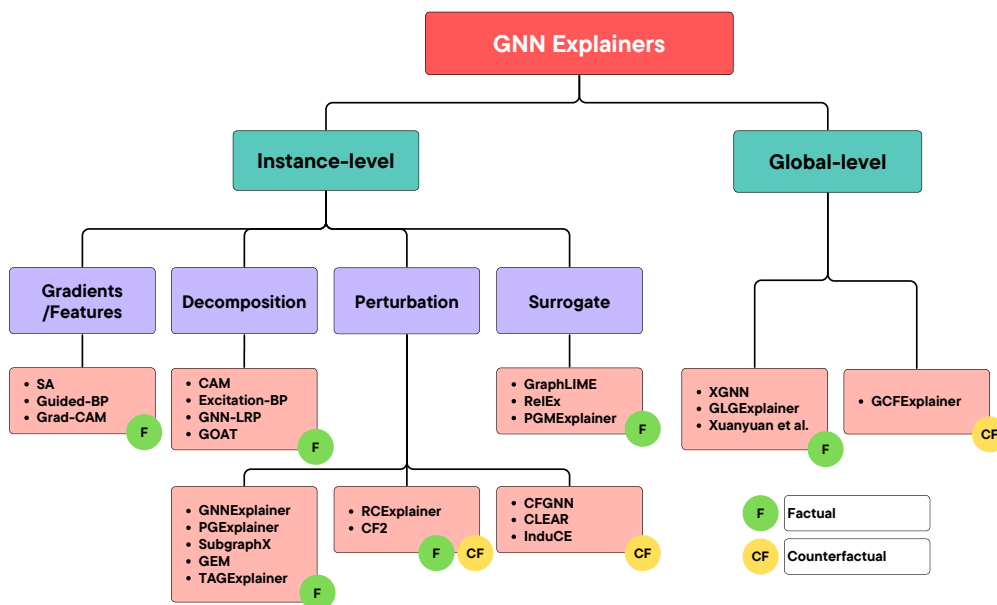

Figure G: Taxonomy of research conducted on explaining GNN predictions. **Gradient:** SA [4] , Guided-BP [4] , Grad-CAM [37]; **Decomposition:** Excitation-BP [37], GNN-LRP [40], CAM [37], GOAT [26]; **Perturbation:** GNNExplainer [55], PGExplainer [29], SubgraphX [58], GEM [25], TAGExplainer [52], CF$^2$ [45], RCExplainer [3],CF-GNNexplainer [27], CLEAR [30], InduCE [47]; **Surrogate:** GraphLime [15], Relex [60], PGM-Explainer [48]; **Global:** XGNN [56], GLG-Explainer [2], Xuanyuan et al. [54], GCFExplainer [16].

## A  Rooted Tree Isomorphism

To identify the set of unique computation trees GRAPHTRAIL performs rooted tree-isomorphism. An efficient method for this task involves using canonical labels. The canonical label of a graph $\mathcal{G}$ is a unique representation that remains unchanged under isomorphism, i.e., two graphs $\mathcal{G}_1$ and $\mathcal{G}_2$ have the same canonical label if and only if they are isomorphic. We construct the canonical label of a rooted computation tree through Depth-First Canonical Labeling (DFCF) [7]. From a labelled rooted unordered tree we can derive many labelled rooted ordered trees as shown in Fig. H. There is a one-to-one correspondence between a labelled rooted ordered tree and its depth-first string encoding. The ordering of the strings orders the trees and the minimum in the ordering is the canonical label. Each string is a depth-first (preorder) traversal that uses $ to represent a backtrack and # to represent

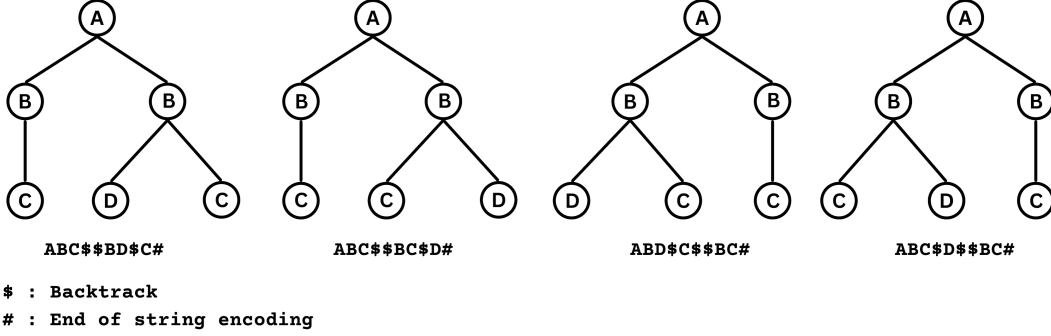

**$ : Backtrack**

**# : End of string encoding**

Figure H: Four rooted ordered trees obtained from the same rooted unordered tree. The alphabets denote the node labels. The strings below each tree are their depth-first traversals.

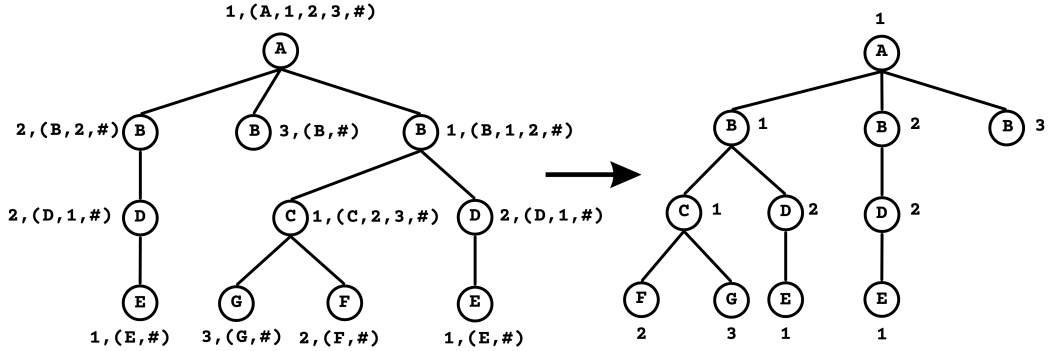

Figure I: Example of constructing a Depth-First-Canonical-Form of a rooted tree. Each vertex is labeled with its identifier and, within parentheses, the ranks of its children, ending with a "hash symbol" to denote the end of the encoding. The rank of the vertex at its level is placed before the parentheses. After sorting all levels, the tree is then scanned from the top down, beginning with the root, and the children of each vertex are rearranged according to the established order.

the end of the string encoding. In sorting, # is greater than $ and both these symbols are greater than other labels. The Depth-First Canonical Form is constructed by sorting the vertices of a rooted unordered tree level by level. At each level, the vertices are sorted first by their labels and then by the ranks of their children at respective levels. Fig. I illustrates the process.

## B  Shapley Values

SHAP [28] represents the Shapley value explanation (SV) [43] as an additive feature attribution method and fits a weighted linear regression model to approximate the model's output. The weights in this model correspond to the SVs (contribution of features). It specifies the explanation as

$$g(z) = \phi_0 + \sum_{j=1}^{M} \phi_j z_j$$

where $g$ is the model, $z \in \{0,1\}^M$ is the coalition vector, $M$ is the maximum coalition size, and $\phi_j \in \mathbb{R}$ is the SV for feature $j$. Kernel-SHAP [28] is a SHAP variant to estimate SVs. It calculates SVs via:

- **Coalition Sampling:** It randomly selects subsets of features, coalitions $(z_k)$. It assigns weights to each coalition using the SHAP kernel where $M$ = total features, $|z|$ = coalition size:

$$\pi(z) = \frac{(M-1)}{\binom{M}{x}|z|(M-|z|)}$$

- **Predictions:** For each coalition, Kernel-SHAP maps the coalition back to the original features and obtains a prediction afterwards.

- **Fitting:** Using the calculated weights and predictions, Kernel-SHAP fits a linear model.

## C Symbolic Regression

In symbolic regression, given a set of $n$ input-output pairs $(x_i, y_i)_{i=1}^n$, where $x_i \in \mathbb{R}^d$ is an input vector, and $y \in \mathbb{R}$ is the output value (or label), and a set of operators, such as addition, subtraction, etc., the goal is to find a symbolic equation $e$ and corresponding function $f_e$ such that $\forall i : y_i \approx f_e(x_i)$, while also reducing the complexity (number of operators and variable) of the formula. The loss function is presented in Eq. 11. Finding the optimal formula is not computationally tractable since the number of formulas grows exponentially with the number of variables and operations. Hence, from the various approximation strategies in the literature [31] we use [8], which leverages an evolutionary algorithm.

The process begins by initializing $n_p$ populations, each with $L$ random expressions of complexity $C$ (3 in our experiments). For each $P_i$, a set $M_i$ is created to store the expressions with the smallest loss (Eq. 11) at each complexity level within that population. Additionally, a global set $H$ is maintained to store the expressions with lowest loss at each complexity across all populations. The algorithm iterates over each population $P_i$ for a fixed number of epochs, allowing them to evolve. Evolution is driven by running tournaments within each population, where the expression $E$ with lowest loss is declared the winner. A copy of the winner is created and chosen for *mutation* or *crossover*, forming $E^*$. Mutating an expression involves repeatedly applying random operations from a set of operations, e.g., replacing/adding operators and variables. Crossover selects the two best expressions, $E_1, E_2$ from the population and swapping random sub-expressions between them, forming $E_1^*, E_2^*$. The newly formed expression(s) replace the oldest expression(s) in the population.

Once evolution within each population is complete, the sets $M_i$ and $H$ are updated with the best expressions. When the set number of epochs are complete, the expression from $H$ with minimum loss is returned.

## D Dataset details

Table C presents four benchmark datasets used in our experiments. NCI1 [49], MUTAG [17], and Mutagenicity [39, 20] are collections of molecules. The nodes represent different atoms, and the edges are chemical bonds between them. The molecules are classified by whether they are anticancer (NCI1) or mutagenic (MUTAG and Mutagenicity). In MUTAG, Class 1 is the mutagenic class while in Mutagenicity, Class 0 is mutagenic. BAMultiShapes [55] is a synthetic dataset. The dataset is composed of 1,000 Barabasi-Albert (BA) graphs with network motifs attached in random positions. These motifs include *house*, *grid*, and *wheel* structures. Class 0 includes plain BA graphs as well as BA graphs augmented with a house, a grid, a wheel, or all three motifs combined. Class 1 consists of BA graphs enriched with combinations of two motifs: a house and a grid, a house and a wheel, or a wheel and a grid.

|  | BAMultiShapes | Mutagenicity | MUTAG | NCI1 |
|---|---|---|---|---|
| #Graphs | 1000 | 4337 | 188 | 4110 |
| Avg. $|\mathcal{V}|$ | 40 | 30.32 | 17.90 | 29.87 |
| Avg. $|\mathcal{E}|$ | 87.00 | 30.77 | 39.60 | 32.30 |
| #Node features | 10 | 14 | 7 | 37 |

Table C: Statistics of the datasets.

## E Further details of empirical framework

All experiments are performed on an Intel Xeon Gold 6248 processor with 96 cores and 1 NVIDIA A100 GPU with 40GB of memory and 377 GB RAM with Ubuntu 18.04.

Each dataset is split into train-validation-test sets in the proportion of 70:10:20. The accuracies of all GNN architectures are reported in Table D. All GNNs have been trained with $L = 3$ layers. We use Adam optimizer with a learning rate set to 0.001. Training stops early after a warm-up of 90

epochs if validation accuracy doesn't increase for 100 epochs or a total of 1000 epochs elapse. The explainers are evaluated on the test splits.

| | GCN | | | GIN | | | GAT | | |
|---|---|---|---|---|---|---|---|---|---|
| | **Sum** | **Mean** | **Max** | **Sum** | **Mean** | **Max** | **Sum** | **Mean** | **Max** |
| BAMultiShapes | 0.83 | 0.83 | 0.82 | 0.94 | 0.93 | 0.91 | 0.5 | 0.5 | 0.5 |
| MUTAG | 0.84 | 0.79 | 0.74 | 0.82 | 0.84 | 0.87 | 0.82 | 0.85 | 0.71 |
| Mutagenicity | 0.79 | 0.77 | 0.77 | 0.79 | 0.80 | 0.77 | 0.77 | 0.78 | 0.75 |
| NCI1 | 0.76 | 0.76 | 0.76 | 0.77 | 0.78 | 0.76 | 0.74 | 0.74 | 0.74 |

Table D: Model accuracy across different GNN architectures and pooling layers for aggregating node embeddings into graph embedding (Eq. 4).

```
In [3]:   edge_lists, graph_labels, edge_label_lists, node_label_lists = get_graph_data(args.dataset)
          with open('./dataset/Mutagenicity.pkl','rb') as fin:
              original_adjs,original_features,original_labels = pkl.load(fin)

              # we only consider the mutagen graphs with NO2 and NH2.
          selected = []
          for gid in range(original_adjs.shape[0]):
              if np.argmax(original_labels[gid]) == 0 and np.sum(edge_label_lists[gid]) > 0:
                  selected.append(gid)
          print('number of mutagen graphs with NO2 and NH2',len(selected))
          selected_adjs = original_adjs[selected]
          selected_features = original_features[selected]
          selected_labels = original_labels[selected]
          selected_edge_lists = [edge_lists[i] for i in selected]
          selected_graph_labels=graph_labels[selected]
          selected_edge_label_lists=[edge_label_lists[i] for i in selected]
          selected_node_label_lists=[node_label_lists[i] for i in selected]

          number of mutagen graphs with NO2 and NH2 1015
```

Figure J: Code snippet of instance explainer restricting train set to only those graphs that contain the explanation motifs.

## F    Contribution of Shapley values

To highlight the impact of using Shapley values for selecting the important computation trees, we compare the GRAPHTRAIL's fidelity when the computation trees are chosen based on the Shapley values versus when they are chosen at random. As evident from Table E, choosing random trees significantly deteriorates the fidelity, indicating the value of Shapley-based selection.

## G    Additional Experiments

In this section, we present additional experiments to further validate our findings. The results are summarized in Tables F, G, H, I, each highlighting different aspects of the derived formulae. Each metric is the weighted average across the classes. These tables provide deeper insights and reinforce the conclusions drawn from our initial analysis.

| | **BAMultiShapes** | **MUTAG** | **Mutagenicity** | **NCI1** |
|---|---|---|---|---|
| Random | $0.54 \pm 0.03$ | $0.68 \pm 0.04$ | $0.55 \pm 0.01$ | $0.54 \pm 0.02$ |
| Shapley | $\mathbf{0.86 \pm 0.02}$ | $\mathbf{0.82 \pm 0.09}$ | $\mathbf{0.72 \pm 0.01}$ | $\mathbf{0.72 \pm 0.02}$ |

Table E: The impact of selecting computation trees by considering Shapley values versus random selection.

| Method | BAMultiShapes | MUTAG | Mutagenicity | NCI1 |
|---|---|---|---|---|
| GLG | $0.39 \pm 0.11$ | $0.70 \pm 0.05$ | $0.64 \pm 0.03$ | $0.49 \pm 0.16$ |
| G-Trail | $\mathbf{0.87 \pm 0.01}$ | $\mathbf{0.83 \pm 0.09}$ | $\mathbf{0.74 \pm 0.01}$ | $\mathbf{0.74 \pm 0.01}$ |

Table F: Weighted average precision of the generated formulas against the GNN output

| Method | BAMultiShapes | MUTAG | Mutagenicity | NCI1 |
|---|---|---|---|---|
| GLG | $0.48 \pm 0.02$ | $0.74 \pm 0.10$ | $0.62 \pm 0.03$ | $0.57 \pm 0.04$ |
| G-Trail | $\mathbf{0.86 \pm 0.02}$ | $\mathbf{0.82 \pm 0.09}$ | $\mathbf{0.72 \pm 0.01}$ | $\mathbf{0.72 \pm 0.02}$ |

Table G: Weighted average recall of the generated formulas against the GNN output

| Method | BAMultiShapes | MUTAG | Mutagenicity | NCI1 |
|---|---|---|---|---|
| GLG | $0.41 \pm 0.06$ | $0.68 \pm 0.11$ | $0.61 \pm 0.03$ | $0.49 \pm 0.11$ |
| G-Trail | $\mathbf{0.86 \pm 0.02}$ | $\mathbf{0.82 \pm 0.08}$ | $\mathbf{0.72 \pm 0.02}$ | $\mathbf{0.70 \pm 0.03}$ |

Table H: The weighted average F1-score of the generated formulas. The weighted version accounts for label imbalance and can result in an F1-score that is not between precision and recall.

| Method | BAMultiShapes | MUTAG | Mutagenicity | NCI1 |
|---|---|---|---|---|
| GLG | $0.46 \pm 0.03$ | $0.68 \pm 0.00$ | $0.55 \pm 0.03$ | $0.56 \pm 0.04$ |
| G-Trail | $\mathbf{0.83 \pm 0.03}$ | $\mathbf{0.72 \pm 0.05}$ | $\mathbf{0.65 \pm 0.01}$ | $\mathbf{0.67 \pm 0.02}$ |

Table I: Accuracy of the generated formulae against ground-truth labels

## H   Reproducibility Analysis of GLGEXPLAINER

Our analysis reveals inconsistencies between the findings reported in [2] and our own observations with GLGEXPLAINER. Specifically, the reported results in [2] for Mutagenicity dataset might be inflated due to test data leakage. In this dataset, the presence of NO2 and/or NH2 motifs is indicative of a positive class label [55]. However, GLGEXPLAINER utilizes PGEXPLAINER [29] to generate instance explanations. Within GLGEXPLAINER, these explanations are created by excluding training graphs that lack these motifs (see Fig. J for the relevant code snippet from PGEXPLAINER's repository https://github.com/flyingdoog/PGExplainer/blob/master/MUTAG.ipynb). This approach results in biased explanations that overemphasize specific motifs. When all training graphs are included, GLGEXPLAINER produces a wider variety of explanations, leading to a decrease in its overall performance.

For other datasets, GLGEXPLAINER offers instance explanations but lacks transparency regarding the specific PGEXPLAINER settings used to generate them. Despite employing all recommended settings and extensive parameter tuning for PGEXPLAINER, we were unable to replicate the instance explanations presented by the authors of GLGEXPLAINER. Our email to the authors of GLGEXPLAINER pointing out these inconsistencies did not receive a response.

## I   Cluster Impurity of GLGEXPLAINER

In GLGEXPLAINER, the formulae are vectors that are called prototypes. Prototypes are representatives of clusters of local explanations, and it is assumed that the clusters are *pure*. This essentially

means that most of the elements in the same clusters will be similar. To verify this claim, we evaluate random graphs from each cluster for both classes across different seeds. Figures K, L, M, and N show the results in all four datasets across different seeds. We make two important observations. First, the clusters are not pure, and the graphs are not similar in the same cluster. For instance, in seed 45 of MUTAG (Fig. K), class 0 has four types of graphs among five. Second, across classes, there exist similar graphs, which is undesired. For example, in the same figure, seed 729 shows three graphs that are similar across classes.

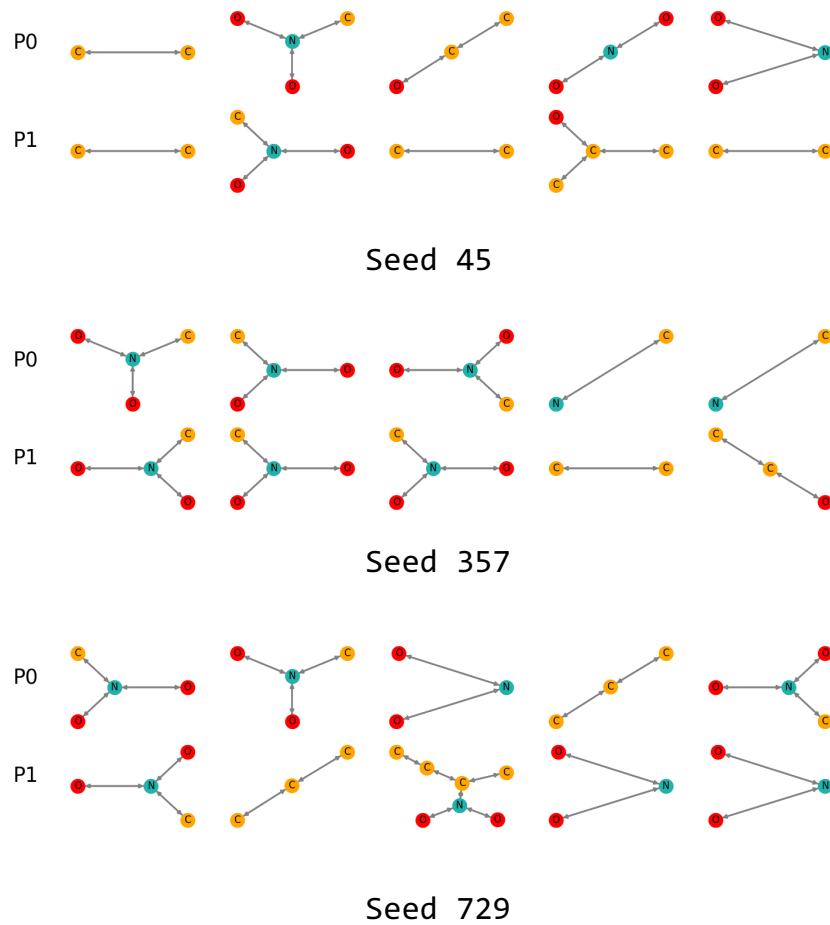

Figure K: A visual depiction of the impure clusters of GLGEXPLAINER in MUTAG.

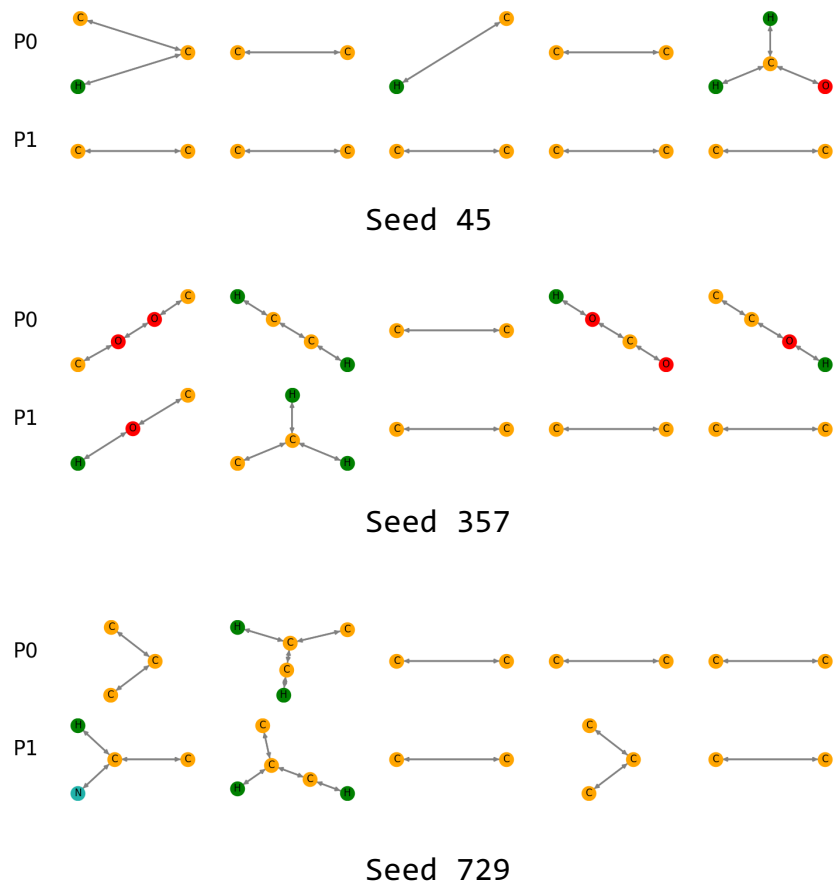

Figure L: A visual depiction of the impure clusters of GLGEXPLAINER in Mutagenicity.

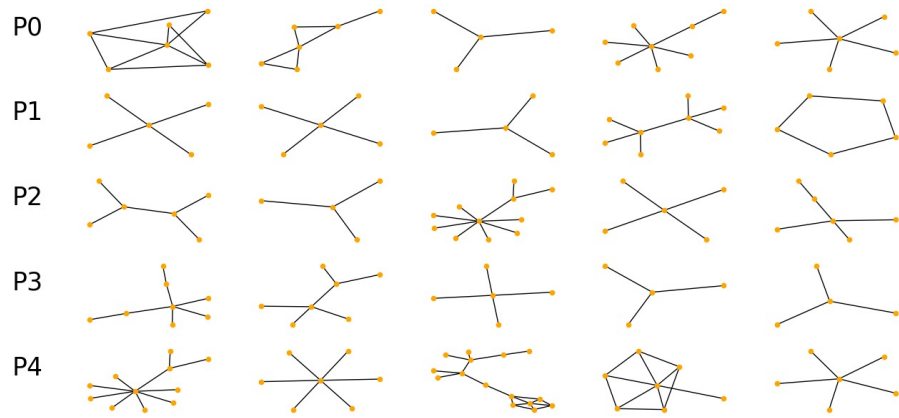

Seed 357

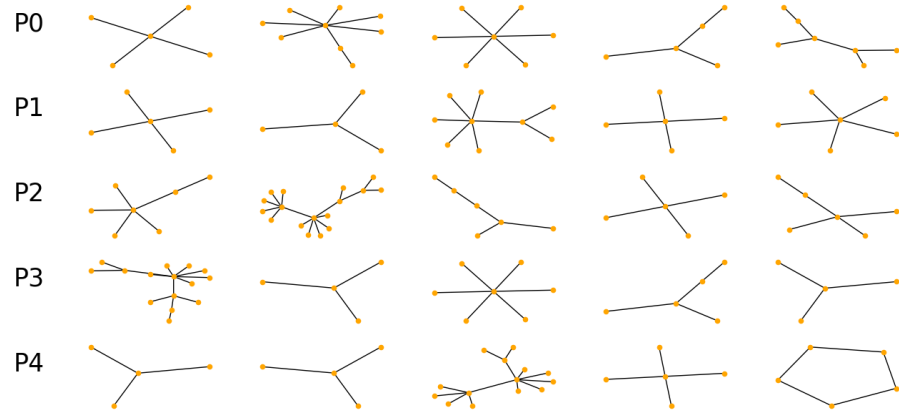

Seed 796

Figure M: A visual depiction of the impure clusters of GLGEXPLAINER in BAMultiShapes.

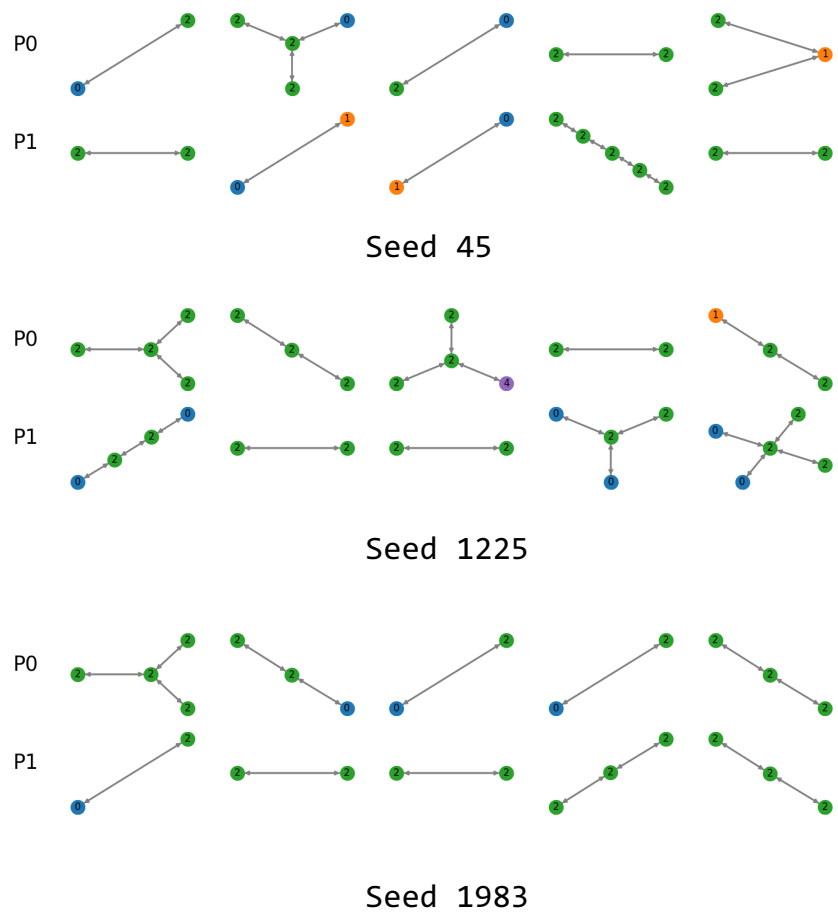

Figure N: A visual depiction of the impure clusters of GLGEXPLAINER in NCI1.

